# GEMINI: GRADIENT ESTIMATION THROUGH MATRIX INVERSION AFTER NOISE INJECTION

Yann Le Cun [1] Conrad C. Galland and Geoffrey E. Hinton
Department of Computer Science
University of Toronto
10 King's College Rd
Toronto, Ontario M5S 1A4
Canada

## ABSTRACT

Learning procedures that *measure* how random perturbations of unit activities correlate with changes in reinforcement are inefficient but simple to implement in hardware. Procedures like back-propagation (Rumelhart, Hinton and Williams, 1986) which *compute* how changes in activities affect the output error are much more efficient, but require more complex hardware. GEMINI is a hybrid procedure for multilayer networks, which shares many of the implementation advantages of correlational reinforcement procedures but is more efficient. GEMINI injects noise only at the first hidden layer and measures the resultant effect on the output error. A linear network associated with each hidden layer iteratively inverts the matrix which relates the noise to the error change, thereby obtaining the error-derivatives. No back-propagation is involved, thus allowing unknown non-linearities in the system. Two simulations demonstrate the effectiveness of GEMINI.

## OVERVIEW

Reinforcement learning procedures typically *measure* the effects of changes in local variables on a global reinforcement signal in order to determine sensible weight changes. This measurement does not require the connections to be used backwards (as in back-propagation), but it is inefficient when more than a few units are involved. Either the units must be perturbed one at a time, or, if they are perturbed simultaneously, the noise from all the other units must be averaged away over a large number of samples in order to achieve a reasonable signal to noise ratio. So reinforcement learning is much less efficient than back-propagation (BP) but much easier to implement in hardware.

GEMINI is a hybrid procedure which retains many of the implementation advantages of reinforcement learning but eliminates some of the inefficiency. GEMINI uses the squared difference between the desired and actual output vectors as a reinforcement signal. It injects random noise at the first hidden layer only, causing correlated noise at later layers. If the noise is sufficiently small, the resultant

change in the reinforcement signal is a linear function of the noise vector at any given layer. A matrix inversion procedure implemented separately at each hidden layer then determines how small changes in the activities of units in the layer affect the reinforcement signal. This matrix inversion gives a much more accurate estimate of the error-derivatives than simply averaging away the effects of noise and, unlike the averaging approach, it can be used when the noise is correlated.

The matrix inversion at each layer can be performed iteratively by a local linear network that "learns" to predict the change in reinforcement from the noise vector at that layer. For each input vector, one ordinary forward pass is performed, followed by a number of forward passes each with a small amount of noise added to the total inputs of the first hidden layer. After each forward pass, one iteration of an LMS training procedure is run at each hidden layer in order to improve the estimate of the error-derivatives in that layer. The number of iterations required is comparable to the width of the largest hidden layer. In order to avoid singularities in the matrix inversion procedure, it is necessary for each layer to have fewer units than the preceding one.

In this hybrid approach, the computations that relate the perturbation vectors to the reinforcement signal are all local to a layer. There is no detailed back-propagation of information, so that GEMINI is more amenable to optical or electronic implementations than BP. The additional time needed to run the gradient-estimating inner loop, may be offset by the fact that only forward propagation is required, so this can be made very efficient (e.g. by using analog or optical hardware).

## TECHNIQUES FOR GRADIENT ESTIMATION

The most obvious way to measure the derivative of the cost function w.r.t the weights is to perturb the weights one at a time, for each input vector, and to measure the effect that each weight perturbation has on the cost function, $C$. The advantage of this technique is that it makes very few assumptions about the way the network computes its output.

It is possible to use far fewer perturbations (Barto and Anandan, 1985) if we are using "quasi-linear" units in which the output, $y_i$, of unit $i$ is a smooth non-linear function, $f$, of its total input, $x_i$, and the total input is a linear function of the incoming weights, $w_{ij}$ and the activities, $y_j$, of units in the layer below:

$$y_i = f(x_i), \qquad x_i = \sum_j w_{ij} y_j$$

Instead of perturbing the weights, we perturb the total input, $x_i$, received by each unit, in order to measure $\partial C / \partial x_i$ . Once this derivative is known it is easy to derive $\partial C / \partial w_{ij}$ for each of the unit's incoming weights by performing a simple *local* computation:

$$\frac{\partial C}{\partial w_{ij}} = \frac{\partial C}{\partial x_i} y_j$$

If the units are perturbed one at a time, we can approximate $\partial C / \partial x_i$ by

$$\frac{\partial C}{\partial x_i} = \frac{\delta C}{\delta x_i} + O(\delta x_i^2)$$

where $\delta C$ is the variation of the cost function induced by a perturbation $\delta x_i$ of the total input to unit $i$. This method is more efficient than perturbing the weights directly, but it still requires as many forward passes as there are hidden units.

### Reducing the number of perturbations required

If the network has a layered, feed-forward, architecture the state of any single layer completely determines the output. This makes it possible to reduce the number of required perturbations and forward passes still further. Perturbing units in the first hidden layer will induce perturbations at the following layers, and we can use these induced perturbations to compute the gradients for these layers. However, since many of the units in a typical hidden layer will be perturbed simultaneously, and since these induced perturbations will generally be correlated, it is necessary to do some local computation within each layer in order to solve the credit assignment problem of deciding how much of the change in the final cost function to attribute to each of the simultaneous perturbations within the layer. This local computation is relatively simple. Let $\mathbf{x}(k)$ be the vector of total inputs to units in layer $k$. Let $\delta\mathbf{x}_t(k)$ be the perturbation vector of layer $k$ at time $t$. It does not matter for the following analysis whether the perturbations are directly caused (in the first hidden layer) or are induced. For a given state of the network, we have:

$$\delta C_t = \frac{\partial C}{\partial \mathbf{x}_k}^T \delta\mathbf{x}_t(k) + O(||\delta\mathbf{x}_t(k)||^2)$$

To compute the gradient w.r.t. layer $k$ we must solve the following system for $\mathbf{g}_k$

$$\delta C_t = \mathbf{g}_k^T \delta\mathbf{x}_t(k) \qquad t = 1 \ldots P$$

where $P$ is the number of perturbations. Unless $P$ is equal to the number of units in layer $k$, *and* the perturbation vectors are linearly independent, this system will be over- or under-determined. In some network architectures it is impossible to induce $n_l$ linearly independent perturbation vectors in a hidden layer, $l$ containing $n_l$ units. This happens when one of the preceding hidden layers, $k$, contains fewer units because the perturbation vectors induced by a layer with $n_k$ units on the following layer generate at most $n_k$ independent directions. So to avoid having to solve an under-determined system, we require "convergent" networks in which each hidden layer has no more units than the preceding layer.

### Using a Special Unit to Allocate Credit within a Layer

Instead of directly solving for the $\partial C/\partial x_i$ within each layer, we can solve the same system iteratively by minimizing:

$$E = \sum_t (\delta C_t - \mathbf{g}_k^T \delta\mathbf{x}_t(k))^2$$

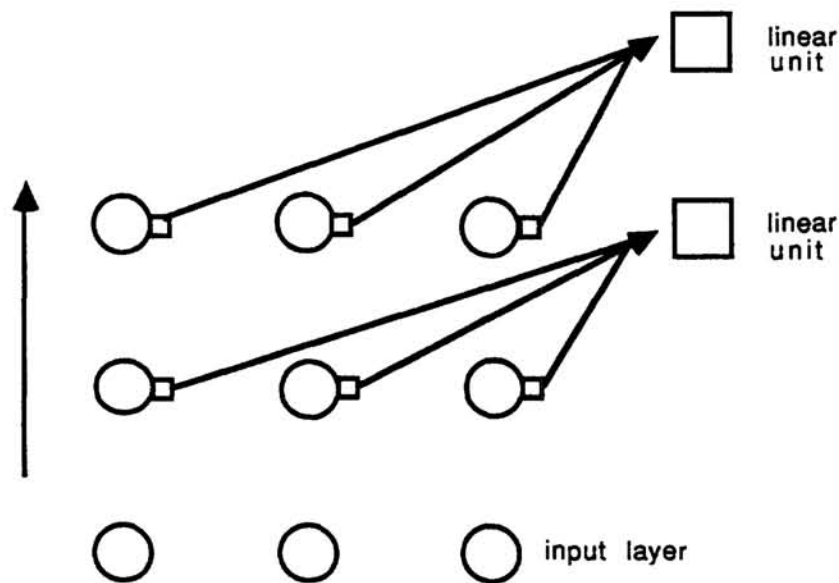

Figure 1: A GEMINI network.

This can be done by a special unit whose inputs are the perturbations of layer $k$ and whose desired output is the resulting perturbation of the cost function $\delta C$ (figure 1). When the LMS algorithm is used, the weight vector $\mathbf{g}_k$ of this special unit converges to the gradient of $C$ with respect to the vector of total inputs $\mathbf{x}(k)$. If the components of the perturbation vector are uncorrelated, the convergence will be fast and the number of iterations required should be of the order of the the number of units in the layer. Each time a new input vector is presented to the main network, the "inner-loop" minimization process that estimates the $\partial C/\partial x_i$ must be re-initialized by setting the weights of the special units to zero or by reloading approximately correct weights from a table that associates estimates of the $\partial C/\partial x_i$ with each input vector.

**Summary of the Gemini Algorithm**

1. Present an input pattern and compute the network state by forward propagation.

2. Present a desired output and evaluate the cost function.

3. Re-initialize the weights of the special units.

4. Repeat until convergence:
   (a) Perturb the first hidden layer and propagate forward.
   (b) Measure the induced perturbations in other layers and the output cost function.
   (c) At each layer apply one step of the LMS rule on the special units to minimize the error between the predicted cost variation and the actual variation.

5. Use the weights of the special units (the estimates of $\partial C/\partial x_i$ ) to compute the weight changes of the main network.

6. Update the weights of the main network.

# A TEST EXAMPLE: CHARACTER RECOGNITION

The GEMINI procedure was tested on a simple classification task using a network with two hidden layers. The input layer represented a 16 by 16 binary image of a handwritten digit. The first hidden layer was an 8 by 8 array of units that were locally connected to the input layer in the following way: Each hidden unit connected to a 3 by 3 "receptive field" of input units and the centers of these receptive fields were spaced two "pixels" apart horizontally and vertically. To avoid boundary effects we used wraparound which is unrealistic for real image processing. The second hidden layer was a 4 by 4 array of units each of which was connected to a 5 by 5 receptive field in the previous hidden layer. The centers of these receptive fields were spaced two pixels apart. Finally the output layer contained 10 units, one for each digit, and was fully connected to the second hidden layer. The network contained 1226 weights and biases.

The sigmoid function used at each node was of the form $f(x) = s\tanh(mx)$ with $m = 2/3$ and $s = 1.716$, thus $f$ was odd, and had the property that $f(1) = 1$ (LeCun, 1987). The training set was composed of 6 handwritten exemplars of each of the 10 digits. It should be emphasized that this task is simple (it is linearly separable), and the network has considerably more weights than are required for this problem.

Experiments were performed with 64 perturbations in the gradient estimation inner loop. Therefore, assuming that the perturbation vectors were linearly independent, the linear system associated with the first hidden layer was not underconstrained [2]. Since a stochastic gradient procedure was used with a single sweep through the training set, the solution was only a rough approximation, though convergence was facilitated by the fact that the components of the perturbations were statistically independent.

The linear systems associated with the second hidden layer and the output layer were almost certainly overconstrained [3], so we expected to obtain a better estimate of the gradient for these layers than for the first one. The perturbations injected at the first hidden layer were independent random numbers with a zero-mean gaussian distribution and standard deviation of 0.1.

The minimization procedure used for gradient estimation was not a pure LMS, but a pseudo-newton method that used a diagonal approximation to the matrix of second derivatives which scales the learning rates for each link independently (Le Cun, 1987; Becker and Le Cun, 1988). In our case, the update rule for a gradient estimate coefficient was

$$g_i \leftarrow g_i + \frac{\eta}{\sigma_i^2}(\delta C - \mathbf{g}^T \delta \mathbf{x})\delta x_i$$

where $\sigma_i^2$ is an estimate of the variance of the perturbation for unit $i$. In the simulations $\eta$ was equal to 0.02 for the first hidden layer, 0.03 for the second hidden layer, and 0.05 for the output layer. Although there was no real need for it, the gradient associated with the output units was estimated using GEMINI so that we could evaluate the accuracy of gradient estimates far away from the noise-injection

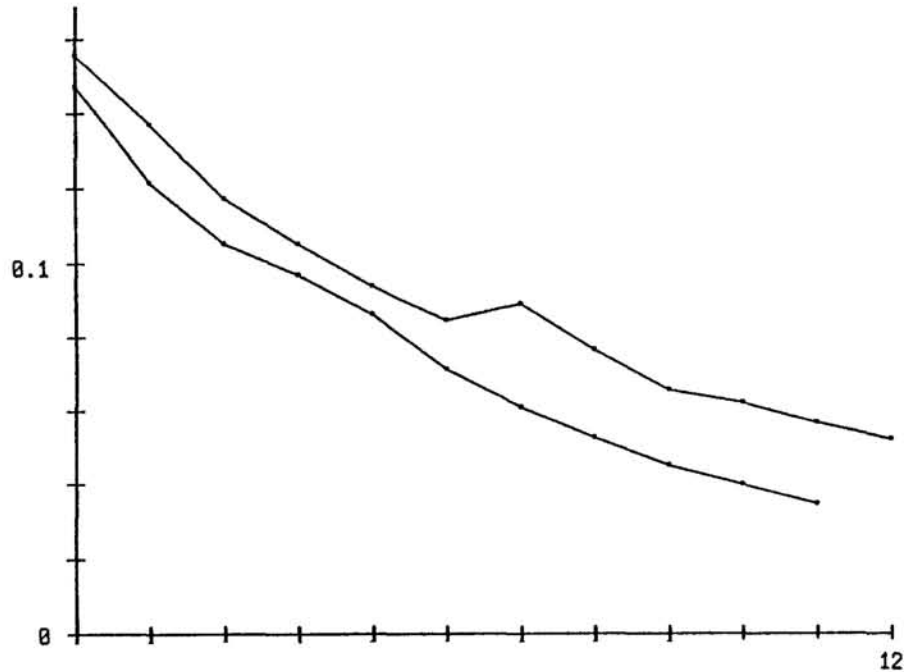

Figure 2: The mean squared error as a function of the number of sweeps through the training set for GEMINI (top curve) and BP (bottom curve).

layer. The learning rates for the main network, $\epsilon_i$, had different values for each unit and were equal to 0.1 divided by the fan-in of the unit.

Figure 2 shows the relative learning rates of BP and GEMINI. The two runs were started from the same initial conditions. Although the learning curve for GEMINI is consistently above the one for BP and is more irregular, the rate of decrease of the two curves is similar. The 60 patterns are all correctly classified after 10 passes through the training set for regular BP, and after 11 passes for GEMINI. In the experiments, the direction of the estimated gradient for a single pattern was within about 20 degrees of the true gradient for the output layer and the second hidden layer, and within 50 degrees for the first hidden layer. Even with such inaccuracies in the gradient direction, the procedure still converged at a reasonable rate.

## LEARNING TO CONTROL A SIMPLE ROBOT ARM

In contrast to the digit recognition task, the robot arm control task considered here is particularily suited to the GEMINI procedure because it contains a non-linearity which is unknown to the network. In this simulation, a network with 2 input units, a first hidden layer with 8 units, a second with 4 units, and an output layer with 2 units is used to control a simulated arm with two angular degrees of freedom. The problem is to train the network to receive x, y coordinates encoded on the two input units and produce two angles encoded on the output units which would place the end of the arm on the desired input point (figure 3). The units use the same input-output function as in the digit recognition example.

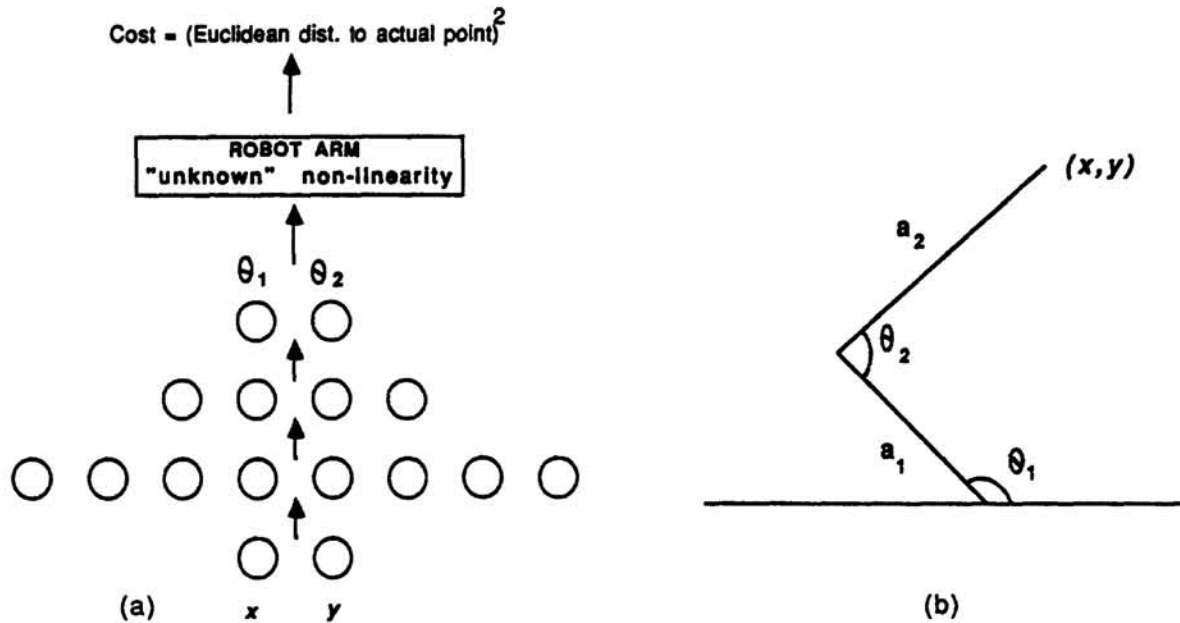

Figure 3: (a) The network trained with the GEMINI procedure, and (b) the 2-D arm controlled by the network.

Each point in the training set is successively applied to the inputs and the resultant output angles determined. The training points are chosen so that the code for the output angles exploits most of the sigmoid input-output curve while avoiding the extreme ends. The "unknown" non-linearity is essentially the robot arm, which takes the joint angles as input and then "outputs" the resulting hand coordinates by positioning itself accordingly. The cost function, $C$, is taken as the square of the Euclidean distance from this point to the desired point. In the simulation, this distance is determined using the appropriate trigonometric relations:

$$C = [(a_1 \cos \theta_1 - a_2 \cos(\theta_1 + \theta_2)) - x]^2 + [(a_1 \sin \theta_1 - a_2 \sin(\theta_1 + \theta_2)) - y]^2$$

where $a_1$ and $a_2$ are the lengths of the two components of the arm. Although this non-linearity is not actually unknown, analytical derivative calculation can be difficult in many real world applications, and so it is interesting to explore the possibility of a control system that can learn without it.

It is found that the minimum number of iterations of the LMS inner loop search needed to obtain good estimates of the gradients when compared to values calculated by back-propagation is between 2 and 3 times the number of units in the first hidden layer (figure 4). For this particular kind of problem, the process can be sped up significantly by using the following two modifications. The same training vector can be applied to the inputs and the weights changed repeatedly until the actual output is within a certain radius of the desired output. The gradient estimates are kept between these weight updates, thereby reducing the number of inner loop

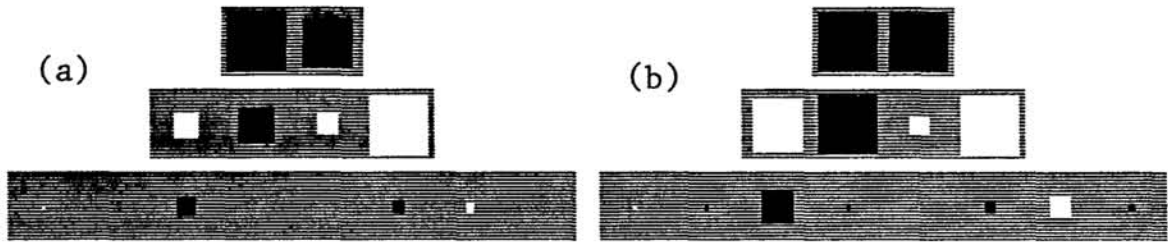

Figure 4: Gradients of the units in all non-input layers, determined (a) by the GEMINI procedure after 24 iterations of the gradient estimating inner loop, and    (b) through analytical calculation. The size of the black and white squares indicates the magnitude of negative and positive error gradients respectively.

iterations needed at each step. The second modification requires that the arm be made to move continuously through 2-D space by using an appropriately ordered training set. The state of the network changes slowly as a result, leading to a slowly varying gradient. Thus, if the gradient estimate is not reset between successive input vectors, it can *track* the real gradient, allowing the number of iterations per gradient estimate to be reduced to as little as 5 in this particular network.

The results of simulations using training sets of closely spaced points in the first quadrant show that GEMINI is capable of training this network to correctly orient the simulated arm, with significantly improved learning efficiency when the above two modifications are employed. Details of these simulation results and the parameters used to obtain them are given in (Galland, Hinton, and Le Cun, 1989).

## Acknowledgements

This research was funded by grants from the Ontario Information Technology Research Center, the Fyssen Foundation, and the National Science and Engineering Research Council. Geoffrey Hinton is a fellow of the Canadian Institute for Advanced Research.

## Footnotes

[1] First Author's present address: Room 4G-332, AT&T Bell Laboratories, Crawfords Corner Rd, Holmdel, NJ 07733

[2] It may have been overconstrained since the actual relation between the perturbation and variation of the cost function is usually non-linear for finite perturbations

[3] This depends on the degeneracy of the weight matrices

## References

A. G. Barto and P. Anandan (1985) Pattern recognizing stochastic learning automata. *IEEE Transactions on Systems, Man and Cybernetics*, **15**, 360–375.

S. Becker and Y. Le Cun (1988) Improving the convergence of back-propagation learning with second order methods. In Touretzky, D. S., Hinton, G. E. and Sejnowski, T. J., editors, *Proceedings of the 1988 Connectionist Summer School*, Morgan Kauffman: Los Altos, CA.

C. C. Galland, G. E. Hinton and Y. Le Cun (1989) Technical Report, *in preparation*.

Y. Le Cun (1987) Modèles Connexionnistes de l'Apprentissage. Doctoral thesis, University of Paris, 6.

D. E. Rumelhart, G. E. Hinton, and R. J. Williams (1986) Learning internal representations by back-propagating errors. *Nature*, **323**, 533–536.